# Automatic Annotation of Everyday Movements

**Deva Ramanan and D. A. Forsyth**
Computer Science Division
University of California, Berkeley
Berkeley, CA 94720
ramanan@cs.berkeley.edu, daf@cs.berkeley.edu

## Abstract

*This paper describes a system that can annotate a video sequence with: a description of the appearance of each actor; when the actor is in view; and a representation of the actor's activity while in view. The system does not require a fixed background, and is automatic. The system works by (1) tracking people in 2D and then, using an annotated motion capture dataset, (2) synthesizing an annotated 3D motion sequence matching the 2D tracks. The 3D motion capture data is manually annotated off-line using a class structure that describes everyday motions and allows motion annotations to be composed — one may jump while running, for example. Descriptions computed from video of real motions show that the method is accurate.*

## 1. Introduction

It would be useful to have a system that could take large volumes of video data of people engaged in everyday activities and produce annotations of that data with statements about the activities of the actors. Applications demand that an annotation system: is wholly automatic; can operate largely independent of assumptions about the background or the number of actors; can describe a wide range of everyday movements; does not fail catastrophically when it encounters an unfamiliar motion; and allows easy revision of the motion descriptions that it uses. We describe a system that largely has these properties. We **track** multiple figures in video data automatically. We then **synthesize** 3D motion sequences matching our 2D tracks using a collection of annotated motion capture data, and then apply the annotations of the synthesized sequence to the video.

**Previous work** is extensive, as classifying human motions from some input is a matter of obvious importance. Space does not allow a full review of the literature; see [1, 5, 4, 9, 13]. Because people do not change in appearance from frame to frame, a practical strategy is to cluster an appearance model for each possible person over the sequence, and then use these models to drive detection. This yields a **tracker** that is capable of meeting all our criteria, described in greater detail in [14]; we used the tracker of that paper. Leventon and Freeman show that tracks can be significantly improved by comparison with human motion [12].

**Describing motion** is subtle, because we require a set of categories into which the motion can be classified; except in the case of specific activities, there is no known natural set of categories. Special cases include ballet and aerobic moves, which have a clearly established categorical structure [5, 6]. In our opinion, it is difficult to establish a canonical set of human motion categories, and more practical to produce a system that allows easy revision of the categories (section 2).

Figure 1 shows an overview of our approach to activity recognition. We use 3 core components; annotation, tracking, and motion synthesis. Initially, a user labels a collection of 3D motion capture frames with annotations (section 2). Given a new video sequence to annotate, we use a kinematic tracker to obtain 2D tracks of each figure in sequence (section 3).

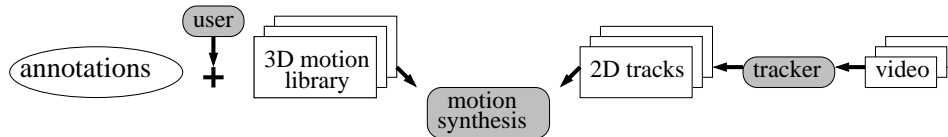

Figure 1: *Our annotation system consists of 3 main components; annotation, tracking, and motion synthesis (the shaded nodes). A* **user** *initially labels a collection of 3D motion capture frames with annotations. Given a new video sequence to annotate, we use a kinematic* **tracker** *to obtain 2D tracks of each figure in sequence. We then* **synthesize** *3D motion sequences which look like the 2D tracks by lifting tracks to 3D and matching them to our annotated motion capture library. We accept the annotations associated with the synthesized 3D motion sequence as annotations for the underlying video sequence.*

We then synthesize 3D motion sequences which look like the 2D tracks by lifting tracks to 3D and matching them to our annotated motion capture library (section 4). We finally smooth the annotations associated with the synthesized 3D motion sequence (section 5), accepting them as annotations for the underlying video sequence.

## 2. Obtaining Annotated Data

We have annotated a body of motion data with an annotation system, described in detail in [3]; we repeat some information here for the convenience of the reader.

There is no reason to believe that a canonical annotation vocabulary is available for everyday motion, meaning that the system of annotation should be flexible. Annotations should allow for composition as one can `wave` while `walking`, for example. We achieve this by representing each separate term in the vocabulary as a bit in a bit string. Our annotation system attaches a bit string to each frame of motion. Each bit in the string represents annotation with a particular element of the vocabulary, meaning that elements of the vocabulary can be composed arbitrarily.

Actual annotation is simplified by using an approach where the user bootstraps a classifier. One SVM classifier is learned for each element of the vocabulary. The user annotates a series of example frames by hand by selecting a sequence from the motion collection; a classifier is then learned from these examples, and the user reviews the resulting annotations. If they are not acceptable, the user revises the annotations at will, and then re-learns a classifier. Each classifier is learned independently. The classifier itself uses a radial basis function kernel, and uses the joint positions for one second of motion centered at the frame being classified as a feature vector. Since the motion is sampled in time, each joint has a discrete 3D trajectory in space for the second of motion centered at the frame. In our implementation, we used a public domain SVM library (`libsvm` [7]). The out of margin cost for the SVM is kept high to force a good fit within the capabilities of the basis function approximation.

Our reference collection consists of a total of 7 minutes of motion capture data. The vocabulary that we chose to annotate this database consisted of: `run`, `walk`, `wave`, `jump`, `turn left`, `turn right`, `catch`, `reach`, `carry`, `backwards`, `crouch`, `stand`, and `pick up`. Some of these annotations co-occur: `turn left` while `walking`, or `catch` while `jumping` and `running`. Our approach admits any combination of annotations, though some combinations may not be used in practice: for example, we can't conceive of a motion that should be annotated with both `stand` and `run`. A different choice of vocabulary would be appropriate for different collections. The annotations are not required to be canonical. We have verified that a consistent set of annotations to describe a motion set can be picked by asking people outside our research group to annotate the same database and comparing annotation results.

## 3. Kinematic Tracking

We use the tracker of [14], which is described in greater detail in that paper. We repeat some information here for the convenience of the reader. The tracker works by building an appearance model of putative actors, detecting instances of that model, and linking the instances across time.

The **appearance model** approximates a view of the body as a puppet built of colored, textured rectangles. The model is built by applying detuned body segment detectors to some or all frames in a sequence. These detectors respond to roughly parallel contrast energies at a set of fixed scales (one for the torso and one for other segments). A detector response at a given position and orientation suggests that there may be a rectangle there. For the frames that are used to build the model, we cluster together segments that are sufficiently close in appearance — as encoded by a patch of pixels within the segment — and appear in multiple frames without violating upper bounds on velocity. Clusters that contain segments that do not move at any point of the sequence are then rejected. The next step is to build assemblies of segments that lie together like a body puppet. The torso is used as a root, because our torso detector is quite reliable. One then looks for segments that lie close to the torso in multiple frames to form arm and leg segments. This procedure does not require a reliable initial segment detector, because we are using many frames to build a model — if a segment is missed in a few frames, it can be found in others. We are currently assuming that each individual is differently dressed, so that the number of individuals is the number of distinct appearance models. **Detecting** the learned appearance model in the sequence of frames is straightforward [8].

## 4. 3D Motion Synthesis

Once the 2D configuration of actors has been identified, we need to synthesize a sequence of 3D configurations matching the 2D reports. Maintaining a degree of smoothness — i.e. ensuring that not only is a 3D representation a good match to the 2D configuration, but also links well to the previous and future 3D representations — is a needed because the image detection is not perfect. We assume that camera motion can be recovered from a video sequence and so we need only to recover the pose of the root of the body model — in our case, the torso — with respect to the camera.

**Representing Body Configuration:** We assume the camera is orthographic and is oriented with the y axis perpendicular to the ground plane, by far the most important case. From the puppet we can compute 2D positions for various key points on the body (we use the left-right shoulder, elbow, wrist, knee, ankle and the upper & lower torso). We represent the 2D key points with respect to a 2D torso coordinate frame. We analogously convert the motion capture data to 3D key points represented with respect to the 3D torso coordinate frame.

We assume that all people are within an isotropic scaling of one another. This means that the scaling of the body can be folded in with the camera scale, and the overall scale is be estimated using corresponding limb lengths in lateral views (which can be identified because they maximize the limb lengths). This strategy would probably lead to difficulties if, for example, the motion capture data came from an individual with a short torso and long arms; the tendency of ratios of body segment lengths to vary from individual to individual and with age is a known, but not well understood, source of trouble in studies of human motion [10].

Our motion capture database is too large for us to use every frame in the matching process. Furthermore, many motion fragments are similar — there is an awful lot of running — so we vector quantize the 11,000 frames down to $k = 300$ frames by clustering with $k$-means and retaining only the cluster medoids. Our distance metric is a weighted sum of differences between 3D key point positions, velocities, and accelerations ([2] found this metric sufficient to ensure smooth motion synthesis). The motion capture data are

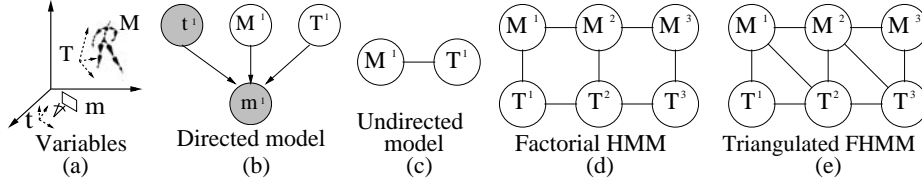

Figure 2: *In **(a)**, the variables under discussion in camera inference. $M$ is a representation of figure in 3D with respect to its root coordinate frame, $m$ is the partially observed vector of 2D key points, $t$ is the known camera position and $T$ is the position of the root of the 3D figure. In **(b)** a camera model for frame $i$ where 2D keypoints are dependent on the camera position, 3D figure configuration, and the root of the 3D figure. A simplified undirected model in **(c)** is obtained by marginalizing out the observed variables yielding a single potential on $M^i$ and $T^i$. In **(d)**, the factorial hidden Markov model obtained by extending the undirected model across time. As we show in the text, it is unwise to yield to the temptation to cut links between $T$'s (or $M$'s) to obtain a simplified model. However, our FHMM is tractable, and yields the triangulated model in **(e)**.*

represented at the same frame rate as the video, to ensure consistent velocity estimates.

**Modeling Root Configuration:** Figure 2 illustrates our variables. For a given frame, we have unknowns $M$, a vector of 3D key points and $T$, the 3D global root position. Known are $m$, the (partially) observed vector of 2D key points, and $t$, the known camera position. In practice, we do not need to model the translations for the 3D root (which is the torso); our tracker reports the $(x, y)$ image position for the torso, and we simply accept these reports. This means that $T$ reduces to a single scalar representing the orientation of the torso along the ground plane. The relative out of image plane movement of the torso (in the z direction) can be recovered from the final inferred $M$ and $T$ values by integration — one sums the out of plane velocities of the rotated motion capture frames.

Figure 2 shows the directed graphical model linking these variables for a single frame. This model can be converted to an undirected model — also shown in the figure — where the observed 2D key points specify a potential between $M_i$ and $T_i$. Write the potential for the $i$th frame as $\psi_{\mathrm{view}_i}(M_i, T_i)$. We wish to minimize image error, so it is natural to use backprojection error for the potential. This means that $\psi_{\mathrm{view}_i}(M_i, T_i)$ is the mean squared error between the visible 2D key points $m_i$ and the corresponding 3D keypoints $M_i$ rendered at orientation $T_i$. To handle left-right ambiguities, we take the minimum error over all left-right assignments. To incorporate higher-order dynamic information such as velocities and accelerations, we add keypoints from the two preceding and two following frames when computing the mean squared error.

We quantize the torso orientation $T_i$ into a total of $c = 20$ values. This means that the potential $\psi_{\mathrm{view}_i}(M_i, T_i)$ is represented by a $c \times k$ table (recall that $k$ is the total number of motion capture *medoids* used, section 4).

We must also define a potential linking body configurations in time, representing the continuity cost of placing one motion after another. We write this potential as $\psi_{\mathrm{link}}(M_i, M_{i+1})$. This is a $k \times k$ table, and we set the $(i, j)$'th entry of this table to be the distance between the $j$'th medoid and the frame following the $i$'th medoid, using the metric used for vector quantizing the motion capture dataset (section 4).

**Inferring Root Configuration:** The model of figure 2-(d) is known as a *factorial hidden Markov model* (FHMM) where observations have been marginalized out and is quite tractable. Exact inference requires triangulating the graph (figure 2-(e)) to make explicit additional probabilistic dependencies [11]. The maximum clique size is now 3, making inference $O(k^2cN)$ (where $N$ is the number of total frames). Furthermore, the triangulation allows us to explicitly define the potential $\psi_{\mathrm{torso}}(M_i, T_i, T_{i+1})$ to capture the dependency

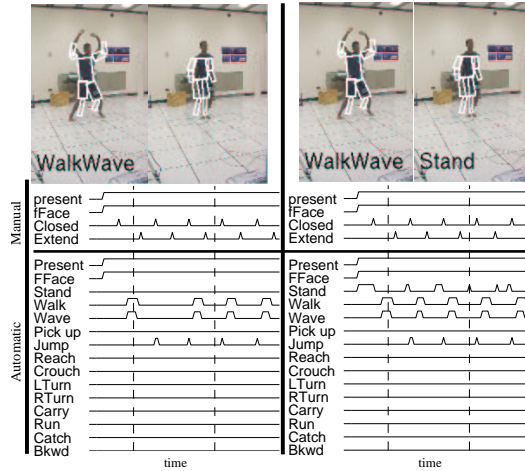

Figure 3: *Unfamiliar configurations can either be annotated with 'null' or with the closest match. We show smoothed annotation results for a sequence of jumping jacks (sometimes known as star jumps) from two such annotation systems. In the top row, we show the same two frames run through each system. The MAP reconstruction of the human figure obtained from the tracking data has been reprojected back to the image, using the MAP estimate of camera configuration. In the bottom, we show signals representing annotation bits over time. The manual annotator records whether or not the figure is* present, front face*ing, in a* closed *stance, and/or in an* extended *stance. The automatic annotation consists of a total of 16 bits;* present, front face*ing, plus the 13 bits from the annotation vocabulary of Sec.2. In first dotted line, corresponding to the image above it, the manual annotator asserts the figure is* present, *frontally* face*ing, and about to reach the* extended *stance. The automatic annotator asserts the figure is* present, *frontally* face*ing, and* walk*ing and* wave*ing, and is* **not** stand*ing,* **not** jump*ing, etc. The annotations for both systems are reasonable given there are no corresponding categories available (this is like describing a movement that is totally unfamiliar). On the* **left**, *we freely allow 'null' annotations (where no annotation bit is set). On the* **right**, *we discourage 'null' annotations as described in Sec.6. Configurations near the* closed *stance are now labeled as* stand*ing, a reasonable approximation.*

of torso angular velocity on the given motion. For example, we expect the torso angular velocity of a turning motion frame to be different from a walking forward frame. We set a given entry of this table to be the squared error between the sampled angular velocity $(T_{i+1} - T_i$, shifted to lie between $-\pi \ldots \pi)$ and the actual torso angular velocity of the medoid $M_i$.

We scale the $\psi_{\text{view}_i}(M_i, T_i)$, $\psi_{\text{link}}(M_i, M_{i+1})$, and $\psi_{\text{torso}}(M_i, T_i, T_{i+1})$ potentials by empirically determined values to yield satisfactory results. These scale factors are weight the degree to which the final 3D track should be continuous versus the degree to which it should match the 2D data. In principle, these weights could be set optimally by a detailed study of the properties of our tracker, but we have found it simpler to set them by experiment.

We find the *maximum a posteriori* (MAP) estimate of $M_i$ and $T_i$ by a variant of dynamic programming defined for clique trees [11]. Since we implicitly used negative log likelihoods to define the potentials (the squared error terms), we used the min-sum variant of the max-product algorithm.

**Possible Variants:** One might choose to not enforce consistency in the root orientation $T_i$ between frames. By breaking the links between the $T_i$ variables in figure 2-(a), we could

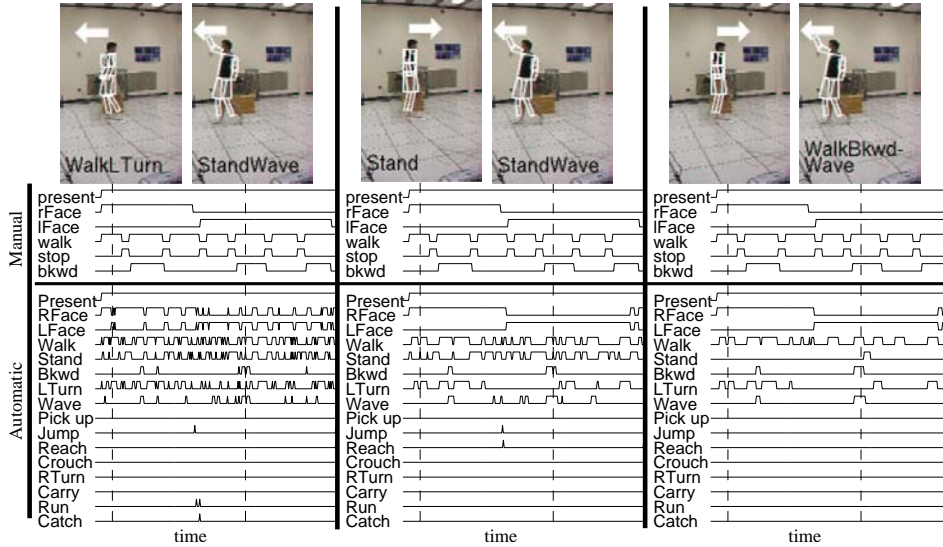

Figure 4: *We show annotation results for a walking sequence from three versions of our system using the notation of Fig.3. Null matches are allowed. On the **left**, we infer the 3D configuration $M^i$ (and associated annotation) independently for each frame, as discussed in Sec.4. In the **center**, we model temporal dependencies when inferring $M^i$ and its corresponding annotation. On the **right**, we smooth the annotations, as discussed in Sec.5. Each image is labeled with an arrow pointing in the direction the inferred figure is facing, not moving. By modeling camera dependencies, we are able to fix incorrect torso orientations present in the **left** system (i.e., the first image frame and the automatic* `left faceing` *and* `right faceing` *annotation bits). By smoothing the annotations, we eliminate spurious* `stand`*'s present in the **center**. Although the smoothing system correctly annotates the last image frame with* `backward`*, the occluded arm incorrectly triggers a* `wave`*, by the mechanism described in Sec.5.*

reduce our model to a tree and make inference even simpler — we now have an HMM. However, this is simplicity at the cost of wasting an important constraint — the camera does not flip around the body from frame to frame. This constraint is useful, because our current image representation provides very little information about the direction of movement in some cases. In particular, in a lateral view of a figure in the stance phase of walking it is very difficult to tell which way the actor is facing without reference to other frames — where it may not be ambiguous. We have found that if one does break these links, the reconstruction regularly flips direction around such frames.

## 5. Reporting Annotations

We now have MAP estimates of the 3D configuration $\{\hat{M}_i\}$ and orientation $\{\hat{T}_i\}$ of the body for each frame. The simplest method for reporting annotations is to produce an annotation that is some function of $\{\hat{M}_i\}$. Recall that $\{\hat{M}_i\}$ is one of the medoids produced by our clustering process (section 4). It represents a cluster of frames, all of which are similar. We could now report either the annotation of the medoid, the annotation that appears most frequently in the cluster, the annotation of the cluster element that matches the image best, or the frequency of annotations across the cluster.

The fourth alternative produces results that may be useful for some kinds of decision-making, but are very difficult to interpret directly — each frame generates a posterior probability over the annotation vocabulary — and we do not discuss it further here. Each of the first three tends to produce choppy annotation streams (figure 4, center). This is because we have vector quantized the motion capture frames, meaning that $\psi_{\text{link}}(M_i, M_{i+1})$ is a

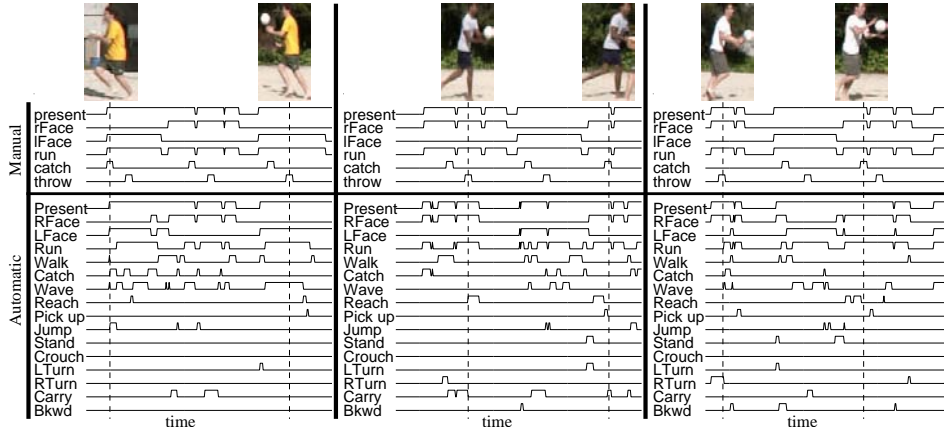

Figure 5: *Smoothed annotations of 3 figures from a video sequence of the three passing a ball back and forth using the conventions of figure 3. Null matches are allowed. The dashed vertical lines indicate annotations corresponding to the frames shown. The automatic annotations are largely accurate: the figures are correctly identified, and the direction in which the figures are facing are largely correct. There is some confusion between* run *and* walk, *and throws appear to be identified as* waves *and* reaches. *Generally, when the figure has the ball (after* catching *and before* throwing, *as denoted in the manual annotations), he is annotated as* carrying, *though there is some false detection. There are no spurious* crouch*es,* turn*s, etc.*

fairly rough approximation of a smoothness constraint (because some frames in one cluster might link well to some frames in another and badly to others in that same cluster). An alternative is to smooth the annotation stream.

**Smoothing Annotations:** Recall that we have 13 terms in our annotation vocabulary, each of which can be on or off for any given frame. Of the $2^{13}$ possible bit strings, we observe a total of 32 in our set of motions. Clearly, we cannot smooth annotation bits directly, because we might very likely create bit strings that never occur. Instead, we regard each observed annotation string as a codeword.

We can model the temporal dynamics of codewords and their quantized observations using a standard HMM. The hidden state is the code word, taking on one of $l$ (= 32) values, while the observed state is the cluster, taking on one of $k$ (= 300) values. This model is defined by a $l \times l$ matrix representing codeword dynamics and a $l \times k$ matrix representing the quantized observation. Note that this model is fully observed in the 11,000 frames of the motion database; we know the true code word for each motion frame and the cluster to which the frame belongs. Hence we can learn both matrices through straightforward multinomial estimation. We now apply this model to the MAP estimate of $\{\hat{M}_i\}$, inferring a sequence of annotation codewords (which we can later expand back into annotation bit vectors).

**Occlusion:** When a limb is not detected by the tracker, the configuration of that limb is not scored in evaluating the potential. In turn, this means that the best configuration consistent with all else detected is used, in this case with the figure waving (figure 4). In an ideal closed world, we can assume the limb is missing because its not there; in practice, it may be due to a detector failure. This makes employing "negative evidence" difficult.

# 6. Experimental Results

It is difficult to evaluate results simply by recording detection information (say an ROC for events). Furthermore, there is no meaningful standard against which one can compare. Instead, we lay out a comparison between human and automatic annotations, as in Fig.3, which shows annotation results for a 91 frame jumping jack (or star jump) sequence. The

top 4 lower case annotations are hand-labeled over the entire 91 frame sequence. Generally, automatic annotation is successful: the figure is detected correctly, oriented correctly (this is recovered from the torso orientation estimates $T_i$), and the description of the figure's activities is largely correct.

Fig.4 compares three versions of our system on a 288 frame sequence of a figure walking back and forth. Comparing the annotations on the left (where configurations have been inferred without temporal dependency) with the center (with temporary dependency), we see temporal dependency in inferred configurations is important, because otherwise the figure can change direction quickly, particularly during lateral views of the stance phase of a walk (section 4). Comparing the center annotations with those on the right (smoothed with our HMM) shows that annotation smoothing makes it possible to remove spurious `jump`, `reach`, and `stand` labels — the label dynamics are wrong.

We show smoothed annotations for three figures from one sequence passing a ball back and forth in Fig.5; the sequence contains a lot of fast movement. Each actor is correctly detected, and the system produces largely correct descriptions of the actor's orientation and actions. The inference procedure interprets a run as a combination of `run` and `walk`. Quite often, the `walk` annotation will fire as the figure slows down to turn from `face right` to `face left` or vice versa. When the figures use their arms to catch or throw, we see increased activity for the similar annotations of `catch`, `wave`, and `reach`.

When a novel motion is encountered, we want the system to either respond by (1) recognizing it cannot annotate this sequence, or (2) annotate it with the best match possible. We can implement (2) by adjusting the parameters for our smoothing HMM so that the 'null' codeword (all annotation bits being off) is unlikely. In Fig.3, system (1) responds to a jumping jack sequence (star jump, in some circles) with a combination of `walking` and `jumping` while `waveing`. In system (2), we see an additional `standing` annotation for when the figure is near the `closed` stance.

## References

[1] J. K. Aggarwal and Q. Cai. Human motion analysis: A review. *Computer Vision and Image Understanding: CVIU*, 73(3):428–440, 1999.
[2] O. Arikan and D. Forsyth. Interactive motion generation from examples. In *Proc. ACM SIGGRAPH*, 2002.
[3] O. Arikan, D. Forsyth, and J. O'Brien. Motion synthesis from annotations. In *Proc. ACM SIGGRAPH*, 2003.
[4] A. Bobick. Movement, activity, and action: The role of knowledge in the perception of motion. *Philosophical Transactions of Royal Society of London*, B-352:1257–1265, 1997.
[5] A. F. Bobick and J. Davis. The recognition of human movement using temporal templates. *IEEE T. Pattern Analysis and Machine Intelligence*, 23(3):257–267, 2001.
[6] L. W. Campbell and A. F. Bobick. Recognition of human body motion using phase space constraints. In *ICCV*, pages 624–630, 1995.
[7] C. C. Chang and C. J. Lin. Libsvm: Introduction and benchmarks. Technical report, Department of Computer Science and Information Engineering, National Taiwan University, 2000.
[8] P. Felzenschwalb and D. Huttenlocher. Efficient matching of pictorial structures. In *Proc CVPR*, 2000.
[9] D. M. Gavrila. The visual analysis of human movement: A survey. *Computer Vision and Image Understanding: CVIU*, 73(1):82–98, 1999.
[10] J. K. Hodgins and N. S. Pollard. Adapting simulated behaviors for new characters. In *SIGGRAPH - 97*, 1997.
[11] M. I. Jordan, editor. *Learning in Graphical Models*. MIT Press, Cambridge, MA, 1999.
[12] M. Leventon and W. Freeman. Bayesian estimation of 3D human motion from an image sequence. Technical Report TR-98-06, MERL, 1998.
[13] D. Ramanan and D. A. Forsyth. Automatic annotation of everyday movements. Technical report, UCB//CSD-03-1262, UC Berkeley, CA, 2003.
[14] D. Ramanan and D. A. Forsyth. Finding and tracking people from the bottom up. In *Proc CVPR*, 2003.
